# Dataflow Architectures: Flexible Platforms for Neural Network Simulation

**Ira G. Smotroff**
MITRE-Bedford Neural Network Group
The MITRE Corporation
Bedford, MA 01730

## ABSTRACT

Dataflow architectures are general computation engines optimized for the execution of fine-grain parallel algorithms. Neural networks can be simulated on these systems with certain advantages. In this paper, we review dataflow architectures, examine neural network simulation performance on a new generation dataflow machine, compare that performance to other simulation alternatives, and discuss the benefits and drawbacks of the dataflow approach.

## 1 DATAFLOW ARCHITECTURES

Dataflow research has been conducted at MIT (Arvind & Culler, 1986) and elsewhere (Hiraki, et. al., 1987) for a number of years. Dataflow architectures are general computation engines that treat each instruction of a program as a separate task which is scheduled in an asynchronous, data-driven fashion. Dataflow programs are compiled into graphs which explicitly describe the data dependencies of the computation. These graphs are directly executed by the machine. Computations which are not linked by a path in the graphs can be executed in parallel. Each machine has a large number of processing elements with hardware that is optimized to reduce task switching overhead to a minimum. As each computation executes and produces a result, it causes all of the following computations that require the result to be scheduled. In this manner, fine grain parallel computation is achieved, with the limit on the amount of possible parallelism determined by the problem and the number of processing elements in the machine.

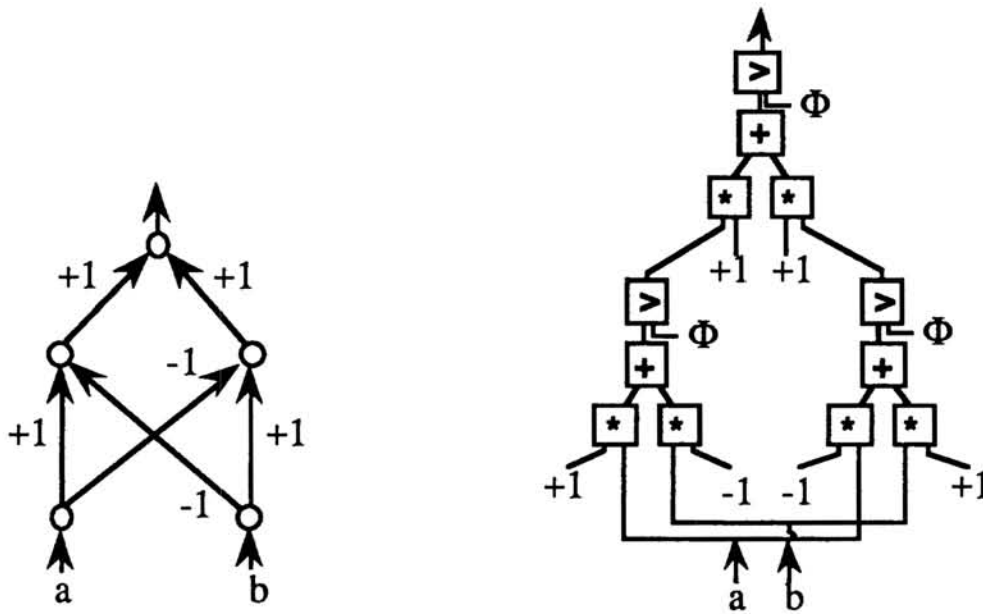

Figure 1:  XOR network and its dataflow graph.

## 1.1 NEURAL NETWORKS & DATAFLOW

The most powerful hardware platforms for neural network simulation were enumerated in the DARPA Neural Network Study (Lincoln Laboratory, 1988): *Supercomputers* offer programming in sequential languages at great cost. *Systolic Arrays* such as the CMU WARP (Pomerleau, 1988) and *"Massively" Parallel* machines such as the Connection Machine (Hillis, 1987), offer power at increasingly reasonable costs, but require specialized low-level programming to map the algorithm to the hardware. *Specialized VLSI* and *Optical devices* (Alspector, 1989) (Farhat, 1987) (Rudnick & Hammerstrom, 1989) offer fast implementations of fixed algorithms[1].

Although dataflow architectures were not included on the DARPA list, there are good reasons for using them for neural network simulation. First, there is a natural mapping between neural networks and the dataflow graphs used to encode dataflow programs (see Figure 1). By expressing a neural network simulation as a dataflow program, one gains the data synchronization and the parallel execution efficiencies that the dataflow architecture provides at an appropriate fine grain of abstraction.  The close mapping may allow simple compilation of neural network specifications into executable programs. Second, this ease of programming makes the approach extremely *flexible,* so one can get good performance on a new algorithm the first time it is run, without having to spend additional time determining the best way to map it onto the hardware. Thus dataflow simulations may be particularly appropriate for those who develop new learning algorithms or architectures. Third, high level languages are being developed for dataflow machines, providing environments in which neural nets can be combined with standard calculations; this can't be done with much of the specialized neural network hardware. Last, there may be ways to optimize dataflow architectures for neural network simulation.

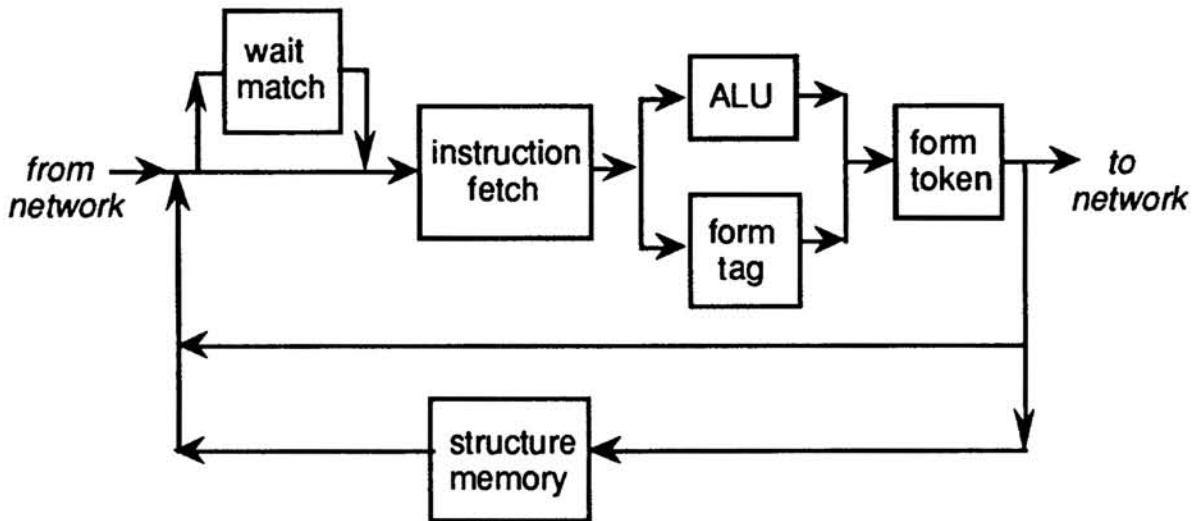

**Figure 2:** Schematic of a tagged-token dataflow processor.

# 2 TAGGED-TOKEN DATAFLOW

The *Tagged-token* dataflow approach represents each computation product as a token which is passed to following computations. A schematic view of a tagged-token processor is shown in Figure 2. Execution proceeds in a *Wait-Match-Store* cycle which achieves data synchronization. An instruction to be executed *waits* in the wait-match queue for a token with its operand. If a *match* occurs, the incoming token contains its operand and one of two things happens: for a monadic operation, the instruction is executed and the result is passed on; for a dyadic operation, a check is made to see if the operand is the first or the second one to arrive. If it's the first, the location representing the instruction is *tagged*, the operand is *stored*, and the instruction continues to wait. If it's the second (i.e. the instruction is tagged already) the instruction is executed and a token containing the result is sent to all computations requiring the result. A schematic view of the execution of the XOR network of Figure 1 on a tagged-token dataflow machine is illustrated in Figure 3.

## 2.1 SPLIT-PHASE TRANSACTIONS

In fine-grain parallel computations distributed over a number of physical devices, the large number of network transactions represent a potential bottleneck. The tagged-token dataflow architecture mitigates this problem in a way that *enhances* the overall parallel execution time. Each network transaction is split into two phases. A process requests an external data value and then goes to sleep. When the token bearing the requested value returns, the process is awakened and the computation proceeds. In standard approaches, a processor must idle while it waits for a result. This non-blocking approach allows other computations to proceed while the value is in transit, *thus masking memory and network latencies*. Independent threads of computation may be interwoven at each cycle, thus allowing the maximum amount of parallel execution at each cycle. As long as the amount of parallelism in the task (i.e. the length of each processor's task queue) is larger than the network latency, the processors never idle. Consequently, massively parallel applications such as neural simulations benefit most from the split-phase transaction approach.

## 3 NEURAL NETWORK DATAFLOW SIMULATION

To illustrate neural network execution on a dataflow processor, the XOR network in Figure 1 was coded in the dataflow language ID (Nikhil, 1988) and run on the MIT GITA (Graph Interpreter for Tagged-token Architecture) simulator (Nikhil, 1988). Figures 4-6 are ALU operations profiles with the vertical axis representing the number of processors that could be simultaneously kept busy (i.e. the amount of parallelism in the task at a particular instance) and the horizontal axis representing elapsed computation cycles. In addition, Figures 4 & 5 are *ideal* simulations with communication latency of zero time and an infinite number of processors available at all times. The ideal profile width represents the absolute minimum time in which the dataflow calculation could possibly be performed, and is termed the *critical path*. Figure 4 shows the execution profile for a single linear threshold neuron processing its two inputs. The initial peak activity of eleven corresponds to initialization activities, with later peaks corresonding to actual computation steps. The complexity of the profile may be attributed to various dataflow synchronization mechanisms. In figure 5, the ideal execution profile for the XOR net, note the initialization peak similar to the one appearing in the single neuron profile; the peak parallelism of fifty-five corresponds to all five neuron initializations occuring simultaneously. This illustrates the ability of the dataflow approach to automatically expose the inherent parallelism in the overall computation. Note also that the critical path of one hundred fifty one is substantially less than five times the single neuron critical path of eighty-five. Wherever possible, the dataflow approach has performed computation in parallel, and the lengthening of the critical path can be attributed to those computations which had to be delayed until prior computations became available.

Figure 6 represents the execution of the same XOR net under more realistic conditions in which each token operation is subject to a finite network delay. The regular spacing of the profile corresponds to the effect of the network delays. The interesting thing to observe is that the overall critical path length has only increased slightly to one hundred seventy because the average amount of parallelism available as tokens come in from the net is higher. Dataflow's ability to interleave computations thus compensates for much of the network latency effects.

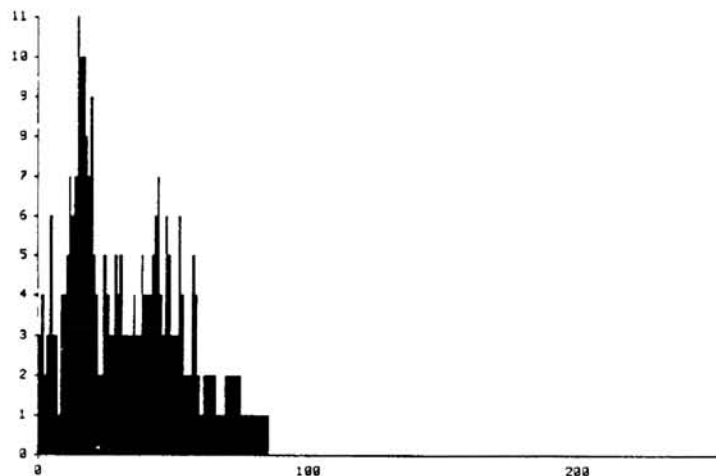

**Figure 4:** Ideal parallelism profile for dataflow execution – single threshold neuron unit.

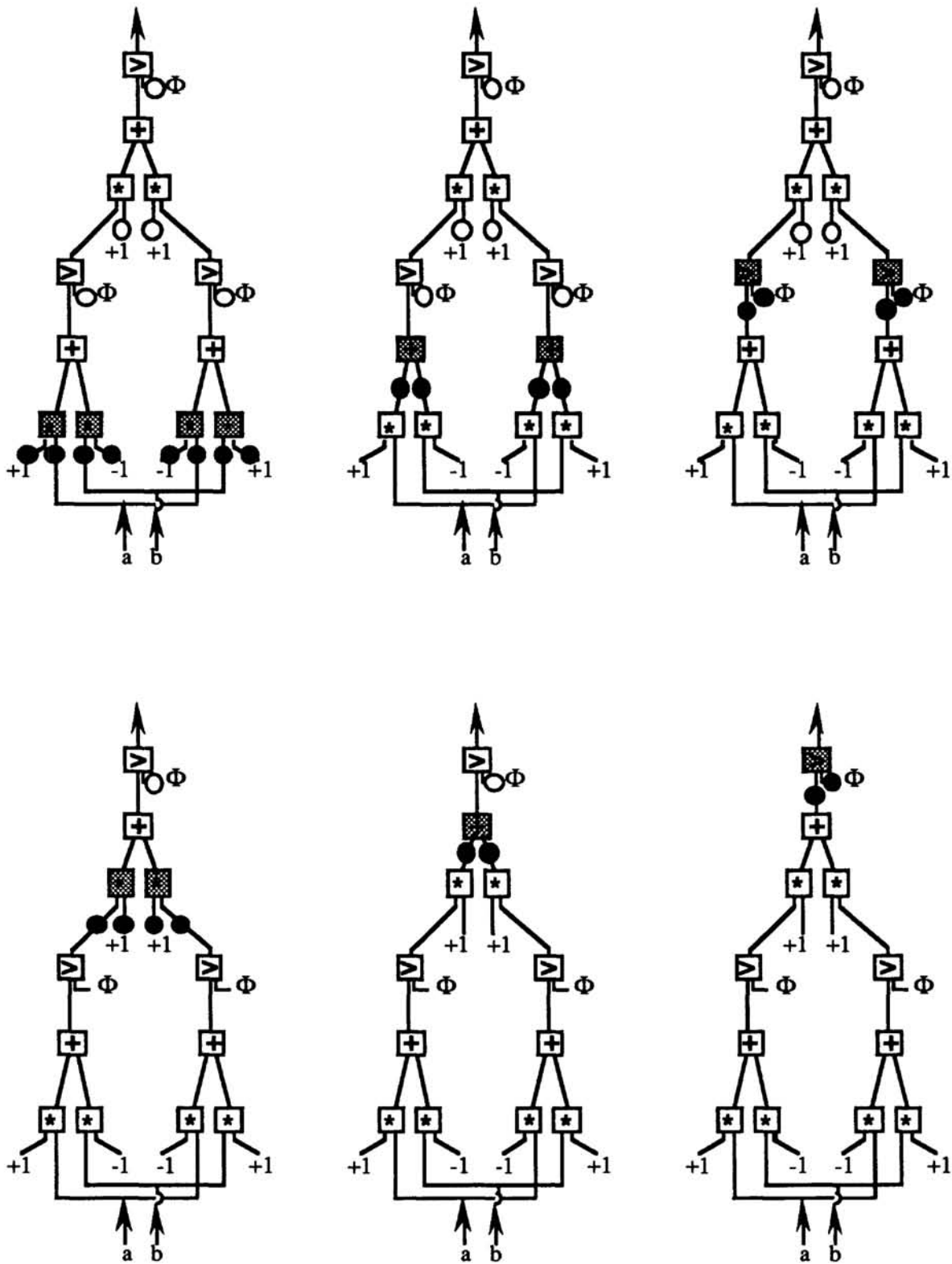

**Figure 3:** Execution of the XOR network of Figure 1 on a tagged-token dataflow processor. The black dots represent active tokens, the white dots represent waiting tokens, and the shaded boxes represent enabled operations executing.

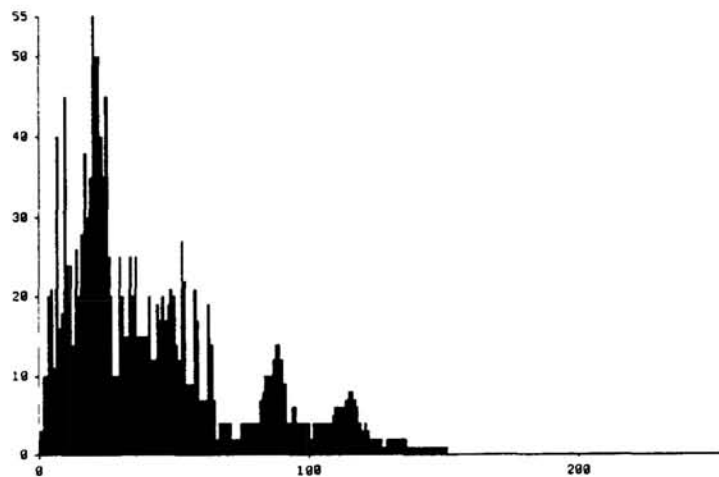

**Figure 5**: Ideal parallelism profile for dataflow execution of XOR network.

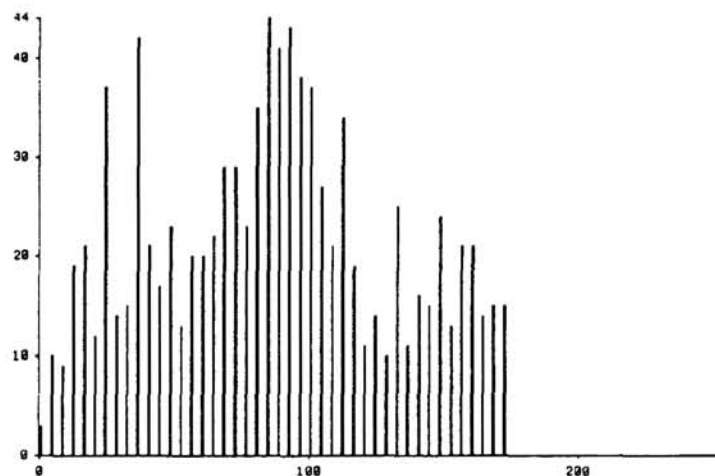

**Figure 6**: Parallelism profile for dataflow execution of XOR with constant communication latency.

## 3.1 COST OF THE DATAFLOW APPROACH

The Tagged-Token Dataflow machine executing an ID program performs two to three times as many instructions as an IBM 370 executing an equivalent FORTRAN program. The overhead in dataflow programs is attributable to mechanisms which manage the asynchronous parallel execution. Similar overhead would probably exist in specialized neural network simulators written for dataflow machines. However, this overhead can be justified because the maximum amount of parallelism in the computation is exposed in a straightforward manner, which requires no additional programming effort. On conventional multiprocessors, parallelism must be selectively tailored for each problem. As the amount of parallelism increases, the associated costs increase as well; often they will eventually surpass the cost of dataflow (Arvind ,Culler & Ekanadham, 1988). Thus the parallel performance on the dataflow machine will often surpass that of alternative platforms despite the overhead.

## 4 THE MONSOON ARCHITECTURE

Early dataflow implementations using a Tagged Token approach had a number of practical barriers (Papadoupoulos, 1988). While useful results were achieved, the cost and expansion limits of the associative memory used for token matching made them impractical. However, the systems did prove the utility of the Tagged Token approach..

Recently, the MONSOON architecture (Papadoupoulos, 1988) was developed to remedy the problems encountered with Tagged Token architectures. The token-matching problem has been solved by treating each token descriptor as an address in a global memory space which is partitioned among the processors in the system; matching becomes a simple RAM operation.

An initial MONSOON prototype has been constructed and a 8 processor machine is scheduled to be built in 1990. Processor elements for that machine are CMOS gate-array implementations being fabricated by Motorola. Each processor board will have a 100 ns cycle time and process at a rate of 7-8 MIPS/2-4 MFLOPS. Total memory for the 8 processor machine is 256 MBytes. Interconnect is provided by a 100 MByte/s packet switch network. The throughput of the 8 processor machine is estimated at 56-64 MIPS/16-32 MFLOPs. This translates to 2-3 million connections per second per processor and 16-24 million connections per second for the machine. Monsoon performance is in the supercomputer class while the projected Monsoon cost is significantly less due to the use of standard process technologies.

A 256 processor machine with CMOS VLSI processors is envisioned. Estimated performance is 40 MIPS per processor and 10,240 MIPS for the machine. Aggregate neural simulation performance is estimated at 2.5-3.8 billion connections per second, assuming an interconnect network of suitable performance.

## 5 CONCLUSIONS

i)   Dataflow architectures should be cost effective and flexible platforms for neural network simulation if they become widely available.

ii)  As general architectures, their performance will not exceed that of specialized neural network architectures.

iii) Maximum parallelism is attained simply by using the dataflow approach: no machine or problem-specific tuning is needed. Thus dataflow is seen as an excellent tool for empirical simulation. Excellent performance may be obtained on cost effective hardware, with no special effort required for performance improvement.

iv)  Dataflow architectures optimized for neural network simulation performance may be possible.

## Footnotes

[1] Hammerstrom's device (Rudnick & Hammerstrom, 1989) may be micro-programmable.

### References

Alspector, J., Gupta, B. and Allen, R. B. (1989) Performance of a Stochastic Learning Microchip. In D. S. Touretzky (ed.), *Advances in Neural Information Processing Systems 1*, 748-760, San Mateo, CA: Morgan Kaufmann.

Arvind and Culler, D. E..(1986) *Dataflow Architectures*, MIT Technical Report MIT/LCS/TM-294, Cambridge, MA.

Arvind, Culler, D. E., Ekanadham, K. (1988) *The Price of Asynchronous Parallelism: An Analysis of Dataflow Architectures*, MIT Laboratory for Computer Science, Computation Structures Group Memo 278.

DARPA Neural Network Study (1988) Lincoln Laboratory, MIT, Lexington, MA.

Farhat, N.H., and Shai, Z. Y.(1987) Architectures and Methodologies for Self-Organization and Stochastic Learning in Opto-Electronic Analogs of Neural Nets. In *Proceedings of IEEE First International Conference on Neural Networks*, **III**:565-576.

Hillis, W. D.(1986) *The Connection Machine*, Cambridge, MA: The MIT Press.

Hiraki, K., Sekiguchi, S. and Shimada, T. (1987) *System Architecture of a Dataflow Supercomputer*. Technical Report, Computer Systems Division, Electrotechnical Laboratory, 1-1-4 Umezono, Sakura-mura, Niihari-gun, Ibaraki, 305, Japan.

Nikhil, R. S. (1988) *Id World Reference Manual*, Computational Structures Group, MIT Laboratory for Computer Science, Cambridge, MA.

Pomerleau, D. A., Gusciora, G. L., Touretsky and D. S., Kung, H. T.(1988) Neural Simulation at Warp Speed: How we got 17 Million Connections per Second. In *Proceedings of the IEEE International Conference on Neural Networks*, **II**:143-150, San Diego.

Papadoupoulos, G. M. (1988) *Implementation of a General Purpose Dataflow Multiprocessor*, Phd. Thesis, MIT Department of Electrical Engineering and Computer Science, Cambridge, MA.

Rudnick, M. and Hammerstrom, D.(1989) An Interconnection Structure for Wafer Scale Neurocomputers. In *Proceedings of the 1988 Connectionist Models Summer School*. San Mateo, CA: Morgan Kaufmann.
